# Accelerated Adaptive Markov Chain
# for Partition Function Computation*

**Stefano Ermon, Carla P. Gomes**
Dept. of Computer Science
Cornell University
Ithaca NY 14853, U.S.A.

**Ashish Sabharwal**
IBM Watson Research Ctr.
Yorktown Heights
NY 10598, U.S.A.

**Bart Selman**
Dept. of Computer Science
Cornell University
Ithaca NY 14853, U.S.A.

## Abstract

We propose a novel Adaptive Markov Chain Monte Carlo algorithm to compute the partition function. In particular, we show how to accelerate a flat histogram sampling technique by significantly reducing the number of "null moves" in the chain, while maintaining asymptotic convergence properties. Our experiments show that our method converges quickly to highly accurate solutions on a range of benchmark instances, outperforming other state-of-the-art methods such as IJGP, TRW, and Gibbs sampling both in run-time and accuracy. We also show how obtaining a so-called *density of states* distribution allows for efficient weight learning in Markov Logic theories.

## 1 Introduction

We propose a novel and general method to approximate the *partition function* of intricate probability distributions defined over combinatorial spaces. Computing the partition function is a notoriously hard computational problem. Only a few tractable cases are know. In particular, if the corresponding graphical model has low treewidth, then the problem can be solved exactly using methods based on tree decompositions, such as the junction tree algorithm [1]. The partition function for planar graphs with binary variables and no external field can also be computed in polynomial time [2].

We will consider an adaptive MCMC sampling strategy, inspired by the Wang-Landau method [3], which is a so-called *flat histogram* sampling strategy from statistical physics. Given a combinatorial space and an energy function (for instance, describing the negative log-likelihood of each configuration), a *flat histogram* method is a sampling strategy based on a Markov Chain that converges to a steady state where it spends approximately the same amount of time in states with a low density of configurations (which are usually low energy states) as in states with a high density.

We propose two key improvements to the Wang-Landau method, namely *energy saturation* and a *focused-random walk* component, leading to a new and more efficient algorithm called `FocusedFlatSAT`. Energy saturation allows the chain to visit fewer energy levels, and the random walk style moves reduce the number of "null moves" in the Markov chain. Both improvements maintain the same global stationary distribution, while allowing us to go well beyond the domain of spin glasses where the Wang-Landau method has been traditionally applied.

We demonstrate the effectiveness of our approach by a comparison with state-of-the-art methods to approximate the partition function or bound it, such as Tree Reweighed Belief Propagation [4], IJGP-SampleSearch [5], and Gibbs sampling [6]. Our experiments show that our approach outperforms these approaches in a variety of problem domains, both in terms of accuracy and run-time.

The *density of states* serves as a rich description of the underlying probabilistic model. Once computed, it can be used to efficiently evaluate the partition function for *all* parameter settings without

the need for further inference steps — a stark contrast with competing methods for partition function computation. For instance, in statistical physics applications, we can use it to evaluate the partition function $Z(T)$ for all values of the temperature $T$. This level of abstraction can be a fundamental advantage for machine learning methods: in fact, in a learning problem we can parameterize $Z(\cdot)$ according to the model parameters that we want to learn from the training data. For example, in the case of a Markov Logic theory [7, 8] with weights $w_1, \ldots, w_K$ of its $K$ first order formulas, we can parameterize the partition function as $Z(w_1, \ldots, w_K)$. Upon defining an appropriate energy function and obtaining the corresponding density of states, we can then use efficient evaluations of the partition function to search for model parameters that best fit the training data, thus obtaining a promising new approach to learning in Markov Logic Networks and graphical models.

## 2   Probabilistic model and the partition function

We focus on intricate probability distributions defined over a set of configurations, i.e., assignments to a set of $N$ discrete variables $\{x_1, \ldots, x_N\}$, assumed here to be Boolean for simplicity. The probability distribution is specified through a set of combinatorial features or *constraints* over these variables. Such constraints can be either *hard* or *soft*, with the $i$-th soft constraint $C_i$ being associated with a weight $w_i$. Let $\chi_i(x) = 1$ if a configuration $x$ violates $C_i$, and 0 otherwise. The probability $P_w(x)$ of $x$ is defined as 0 if $x$ violates any hard constraint, and as

$$P_w(x) = \frac{1}{Z(w)} \exp \left( - \sum_{C_i \in \mathcal{C}_{\text{soft}}} w_i \chi_i(x) \right) \tag{1}$$

otherwise, where $\mathcal{C}_{\text{soft}}$ is the set of soft constraints. The *partition function*, $Z(w)$, is simply the normalization constant for this probability distribution, and is given by:

$$Z(w) = \sum_{x \in \mathcal{X}_{\text{hard}}} \exp \left( - \sum_{C_i \in \mathcal{C}_{\text{soft}}} w_i \chi_i(x) \right) \tag{2}$$

where $\mathcal{X}_{\text{hard}} \subseteq \{0, 1\}^N$ is the set of configurations satisfying all hard constraints. Note that as $w_i \to \infty$, the soft constraint $C_i$ effectively becomes a hard constraint. This factored representation is closely related to a graphical model where we use weighted Boolean formulas to specify clique potentials. This is a natural framework for combining purely logical and probabilistic inference, used for example to define grounded Markov Logic Networks [8, 9].

The partition function is a very important quantity but computing it is a well-known computational challenge, which we propose to address by employing the "density of states" method to be discussed shortly. We will compare our approach against several state-of-the-art methods available for computing the partition function or obtaining bounds on it. Wainwright et al. [4], for example, proposed a variational method known as tree re-weighting (TRW) to obtain bounds on the partition function of graphical models. Unlike standard Belief Propagation schemes which are based on Bethe free energies [10], the TRW approach uses a tree-reweighted (TRW) free energy which consists of a linear combination of free energies defined on spanning trees of the model. Using convexity arguments it is then possible to obtain upper bounds on various quantities, such as the partition function.

Based on iterated join-graph propagation, IJGP-SampleSearch [5] is a popular solver for the *probability of evidence* problem (i.e., partition function computation with a subset of "evidence" variables fixed) for general graphical models. This method is based on an importance sampling scheme which is augmented with systematic constraint-based backtracking search. An alternative approach is to use Gibbs sampling to estimate the partition function by estimating, using sample average, a sequence of *multipliers* that correspond to the ratios of the partition function evaluated at different weight levels [6]. Lastly, the partition function for planar graphs where all variables are binary and have only pairwise interactions (i.e., the zero external field case) can be calculated exactly in polynomial time [2]. Although we are interested in algorithms for the general (intractable) case, we used the software associated with this approach to obtain the ground truth for planar graphs and evaluate the accuracy of the estimates obtained by other methods.

# 3 Density of states

Our approach for computing the partition function is based on solving the density of states problem. Given a combinatorial space such as the one defined earlier and an energy function $E : \{0,1\}^N \to \mathbb{R}$, the *density of states* (DOS) $n$ is a function $n : range(E) \to \mathbb{N}$ that maps energy levels to the number of configurations with that energy, i.e., $n(k) = |\{\sigma \in \{0,1\}^N \mid E(\sigma) = k\}|$. In our context, we are interested in computing the number of configurations that satisfy certain properties that are specified using an appropriate energy function. For instance, we might define the *energy* $E(\sigma)$ of a configuration $\sigma$ to be the number of hard constraints that are violated by $\sigma$. Or we may use the sum of the weights of the violated soft constraints.

Once we are able to compute the full density of states, i.e., the number of configurations at each possible energy level, it is straightforward to evaluate the partition function $Z(w)$ for any weight vector $w$, by summing up terms of the form $n(i) \exp(-E(i))$, where $E(i)$ denotes the energy of every configuration in state $i$. This is the method we use in this work for estimating the partition function. More complex energy functions may be defined for other related tasks, such as *weight learning*, i.e., given some training data $\overline{x} \in \mathcal{X} = \{0,1\}^N$, computing $\arg\max_w P_w(\overline{x})$ where $P_w(\overline{x})$ is given by Equation (1). Here we can define the energy $E(\sigma)$ to be $w \cdot \ell$, where $\ell = (\ell_1, \ldots, \ell_M)$ gives the number of constraints of weight $w_i$ violated by $\sigma$. Our focus in the rest of the paper will thus be on computing the density of states efficiently.

## 3.1 The MCMCFlatSAT algorithm

`MCMCFlatSAT` [11] is an Adaptive Markov Chain Monte Carlo (adaptive MCMC) method for computing the density of states for combinatorial problems, inspired by the Wang-Landau algorithm [3] from statistical physics. Interestingly, this algorithm does not make any assumption about the form or semantics of the energy. At least in principle, the only thing it needs is a partitioning of the state space, where the "energy" just provides an index over the subsets that compose the partition.

The algorithm is based on the flat histogram idea and works by trying to construct a reversible Markov Chain on the space $\{0,1\}^N$ of all configurations such that the steady state probability of a configuration $\sigma$ is inversely proportional to the density of states $n(E(\sigma))$. In this way, the stationary distribution is such that all the energy levels are visited equally often (i.e., when we count the visits to each energy level, we see a flat visit histogram). Specifically, we define a Markov Chain with the following transition probability:

$$p_{\sigma \to \sigma'} = \begin{cases} \frac{1}{N} \min\left\{1, \frac{n(E(\sigma))}{n(E(\sigma'))}\right\} & d_H(\sigma, \sigma') = 1 \\ 0 & d_H(\sigma, \sigma') > 1 \end{cases} \tag{3}$$

where $d_H(\sigma, \sigma')$ is the Hamming distance between $\sigma$ and $\sigma'$. The probability of a self-loop $p_{\sigma \to \sigma}$ is given by the normalization constraint $p_{\sigma \to \sigma} + \sum_{\sigma' \mid d_H(\sigma, \sigma') = 1} p_{\sigma \to \sigma'} = 1$. The detailed balance equation $P(\sigma) p_{\sigma \to \sigma'} = P(\sigma') p_{\sigma' \to \sigma}$ is satisfied by $P(\sigma) \propto 1/n(E(\sigma))$. This means[1] that the Markov Chain will reach a stationary probability distribution $P$ (regardless of the initial state) such that the probability of a configuration $\sigma$ with energy $E = E(\sigma)$ is inversely proportional to the number of configurations with energy $E$. This leads to an asymptotically flat histogram of the energies of the states visited because $P(E) = \sum_{\sigma : E(\sigma) = E} P(\sigma) \propto n(E) \frac{1}{n(E)} = 1$ (i.e., independent of $E$).

Since the density of states is not known a priori, and computing it is precisely the goal of the algorithm, it is not possible to construct directly a random walk with transition probability (3). However it *is* possible to start with an initial guess $g(\cdot)$ for $n(\cdot)$ and keep updating this estimate $g(\cdot)$ in a systematic way to produce a flat energy histogram and simultaneously make the estimate $g(E)$ converge to the true value $n(E)$ for every energy level $E$. The estimate is adjusted using a modification factor $F$ which controls the trade-off between the convergence rate of the algorithm and its accuracy (large initial values of $F$ lead to fast convergence to a rather inaccurate solution). For completeness, we provide the pseudo-code as Algorithm 1; see [11] for details.

**Algorithm 1** `MCMCFlatSAT` algorithm to compute the density of states

---

1: Start with a guess $g(E) = 1$ for all $E = 1, \ldots, m$
2: Initialize $H(E) = 0$ for all $E = 1, \ldots, m$
3: Start with a modification factor $F = F_0 = 1.5$
4: **repeat**
5:     Randomly pick a configuration $\sigma$
6:     **repeat**
7:         Generate a new configuration $\sigma'$ (by flipping a variable)
8:         Let $E = E(\sigma)$ and $E' = E(\sigma')$ (saturated energies)
9:         Set $\sigma = \sigma'$ with probability $\min\left\{1, \frac{g(E)}{g(E')}\right\}$ (move acceptance/rejection step)
10:        Let $E_c = E(\sigma)$ be the current energy level
11:        Adjust the density $g(E_c) = g(E_c) \times F$
12:        Update visit histogram $H(E_c) = H(E_c) + 1$
13:     **until** $H$ is flat (all the values are at least 90% of the maximum value)
14:     Reduce $F$, $F \leftarrow \sqrt{F}$
15:     Reset the visit histogram $H$
16: **until** $F$ is close enough to 1
17: Normalize $g$ so that $\sum_E g(E) = 2^N$
18: **return** $g$ as estimate of $n$

---

## 4   `FocusedFlatSAT`: Efficient computation of density of states

We propose two crucial improvements to `MCMCFlatSAT`, namely *energy saturation* and the introduction of a *focused-random walk* component, leading to a new algorithm called `FocusedFlatSAT`. As we will see in Table 1, `FocusedFlatSAT` provides the same accuracy as `MCMCFlatSAT` but is about 10 times faster on that benchmark. Moreover, our results for the Ising model (described below) in Figure 2 demonstrate that `FocusedFlatSAT` scales much better.

**Energy saturation.** The time needed for each iteration of `MCMCFlatSAT` to converge is significantly affected by the number of different non-empty energy levels (buckets). In many cases, the weights defining the probability distribution $P_w(x)$ are all positive (i.e., there is an incentive to satisfy the constraints), and as an effect of the exponential discounting in Equation (1), configurations that violate a large number of constraints have a negligible contribution to the sum defining the partition function $Z$. We therefore define a new *saturated* energy function $E'(\sigma) = \min\{E(\sigma), K\}$, where $K$ is a user-defined parameter. For the positive weights case, the partition function $Z'$ associated with the saturated energy function is a guaranteed upper bound on the original $Z$, for any $K$. When all constraints are hard, $Z' = Z$ for any value $K \geq 1$ because only the first energy bucket matters. In general, when soft constraints are present, the bound gets tighter as $K$ increases, and we can obtain theoretical worst-case error bounds when $K$ is chosen to be a percentile of the energy distribution (e.g., saturation at median energy yields a 2x bound). In our experiments, we set $K$ to be the average number of constraints violated by a random configuration, and we found that the error introduced by the saturation is negligible compared to other inherent approximations in density of states estimation. Intuitively, this is because the states where the probability is concentrated turn out to typically have a much lower energy than $K$, and thus an exponentially larger contribution to Z. Furthermore, energy saturation preserves the connectivity of the chain.

**Focused Random Walk.** Both in the original Wang-Landau method and in `MCMCFlatSAT`, new configurations are generated by flipping a variable selected uniformly at random [3, 11]. Let us call this configuration selection distribution the *proposal distribution*, and let $T_{\sigma \to \sigma'}$ denote the probability of generating a $\sigma'$ from this distribution while in configuration $\sigma$. In the Wang-Landau algorithm, proposed configurations are then rejected with a probability that depends on the density of states of the respective energy levels. Move rejections obviously lengthen the mixing time of the underlying Markov Chain. We introduce here a novel proposal distribution that significantly reduces the number of move rejections, resulting in much faster convergence rates. It is inspired by local search SAT solvers [12] and is especially critical for the class of highly combinatorial energy functions we consider in this work. We note that if the acceptance probability is taken to be

$$\min\left\{1, \frac{n(E(\sigma))\, T_{\sigma' \to \sigma}}{n(E(\sigma'))\, T_{\sigma \to \sigma'}}\right\}$$

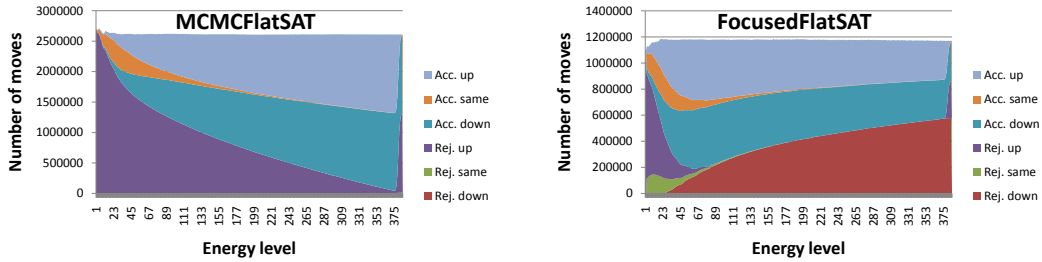

Figure 1: Histograms depicting the number of proposed moves accepted and rejected. Left: MCM-CFlatSAT. Right: FocusedFlatSAT. See PDF for color version.

the properties of the steady state distribution are preserved as long as the proposal distribution is such that the ergodicity property is maintained.

In order to understand the motivation behind the new proposal distribution, consider the move acceptance/rejection histogram shown in the left panel of Figure 1. For the instance under consideration, `MCMCFlatSAT` converged to a flat histogram after having visited each of the 385 energy levels (on x-axis) roughly 2.6M times. Each colored region shows the cumulative number of moves (on y-axis) accepted or rejected from each energy level (on x-axis) to another configuration with a higher, equal, or lower energy level, resp. This gives six possible move types, and the histogram shows how often is each taken at any energy level. Most importantly, notice that at low energy levels, a vast majority of the moves were proposed to a higher energy level and were rejected by the algorithm (shown as the dominating purple region). This is an indirect consequence of the fact that in such instances, in the low energy regime, the density of states increases drastically as the energy level is increases, i.e., $g(E') \gg g(E)$ when $E' > E$. As a result, most of the proposed moves are to higher energy levels and are in turn rejected by the algorithm in the move acceptance/rejection step discussed above.

In order to address this issue, we propose to modify the proposal distribution in a way that increases the chance of proposing moves to the same or lower energy levels, despite the fact that there are relatively few such moves. Inspired by local search SAT solvers, we enhance `MCMCFlatSAT` with a *focused random walk* component that gives preference to selecting variables to flip from violated constraints (if any), thereby introducing an indirect bias towards lower energy states. Specifically, if the given configuration $\sigma$ is a satisfying assignment, pick a variable uniformly at random to be flipped (thus $T_{\sigma \to \sigma'} = 1/N$ when the Hamming distance $d_H(\sigma, \sigma') = 1$, zero otherwise). If $\sigma$ is not a solution, then with probability $p$ a variable to be flipped is chosen uniformly at random from a randomly chosen violated constraint, and with probability $1 - p$ a variable is chosen uniformly at random. With this approach, when $\sigma$ is not solution and $\sigma$ and $\sigma'$ differ only on the $i$-th variable,

$$T_{\sigma \to \sigma'} = (1 - p)\frac{1}{N} + p\frac{\sum_{c \in C | i \in c} \chi_c(\sigma) \cdot 1/|c|}{\sum_{c \in C} \chi_c(\sigma)}$$

where $\chi_c(\sigma) = 1$ iff $\sigma$ violates constraint $c$ and $|c|$ denotes the number of variables in constraint $c$. With this proposal distribution we ensure that for all $1 > p \geq 0$ whenever $T_{\sigma \to \sigma'} > 0$, we also have $T_{\sigma' \to \sigma} > 0$. Moreover, the connectivity of the Markov Chain is preserved (since we don't remove any edge from the original Markov Chain). We therefore have the following result:

**Proposition 1** *For all $p \in [0, 1)$, the Markov Chain with proposal distribution $T_{\sigma \to \sigma'}$ defined above is irreducible and aperiodic. Therefore it has a unique stationary distribution, given by $1/n(E(\sigma))$.*

The right panel of Figure 1 shows the move acceptance/rejection histogram when FocusedFlatSAT is used, i.e., with the above proposal distribution. The same instance now needs under 1.2M visits per energy level for the method to converge. Moreover, the number of rejected moves (shown in purple and green) in low energy states is significantly fewer than the dominating purple region in the left panel. This allows the Markov Chain to move more freely in the space and to converge faster.

Figure 2 shows a runtime comparison of FocusedFlatSAT against `MCMCFlatSAT` on $n \times n$ Ising models (details to be discussed in Section 5). As we see, incorporating energy saturation reduces the time to convergence (while achieving the same level of accuracy), and using focused random walk moves further decreases the convergence time, especially as $n$ increases.

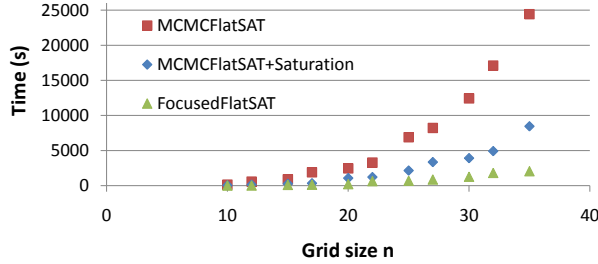

Figure 2: Runtime comparison on ferromagnetic Ising models on square lattices of size $n \times n$.

Table 1: Comparison with model counters; only hard constraints. Runtime is in seconds.

| Instance | n | m | Exact # | FocusedFlatSat Models | Time | MCMC-FlatSat Models | Time | SampleCount Models | Time | SampleMiniSAT Models | Time |
|---|---|---|---|---|---|---|---|---|---|---|---|---|
| 2bitmax_6 | 252 | 766 | $2.10 \times 10^{29}$ | $1.91 \times 10^{29}$ | 156 | $1.96 \times 10^{29}$ | 1863 | $\geq 2.40 \times 10^{28}$ | 29 | $\mathbf{2.08 \times 10^{29}}$ | 345 |
| wff-3-3.5 | 150 | 525 | $1.40 \times 10^{14}$ | $\mathbf{1.43 \times 10^{14}}$ | 20 | $1.34 \times 10^{14}$ | 393 | $\geq 1.60 \times 10^{13}$ | 145 | $1.60 \times 10^{13}$ | 240 |
| wff-3.1.5 | 100 | 150 | $1.80 \times 10^{21}$ | $1.86 \times 10^{21}$ | 1 | $\mathbf{1.83 \times 10^{21}}$ | 21 | $\geq 1.00 \times 10^{20}$ | 240 | $1.58 \times 10^{21}$ | 128 |
| wff-4-5.0 | 100 | 500 | | $9.31 \times 10^{16}$ | 5 | $8.64 \times 10^{16}$ | 189 | $\geq 8.00 \times 10^{15}$ | 120 | $1.09 \times 10^{17}$ | 191 |
| ls8-norm | 301 | 1603 | $5.40 \times 10^{11}$ | $\mathbf{5.78 \times 10^{11}}$ | 231 | $5.93 \times 10^{11}$ | 2693 | $\geq 3.10 \times 10^{10}$ | 1140 | $2.22 \times 10^{11}$ | 168 |

# 5   Experimental evaluation

We compare `FocusedFlatSAT` against several state-of-the-art methods for computing an estimate of or bound on the partition function.[2] An evaluation such as this is inherently challenging as the ground truth is very hard to obtain and computational bounds can be orders of magnitude off from the truth, making a comparison of estimates not very meaningful. We therefore propose to evaluate the methods on either small instances whose ground truth can be evaluated by "brute force," or larger instances whose ground truth (or bounds on it) can be computed analytically or through other tools such as efficient model counters. We also consider planar cases for which a specialized polynomial time exact algorithm is available. Efficient methods for handling instances of small treewidth are also well known; here we push the boundaries to instances of relatively higher treewidth.

For partition function evaluation, we compare against the tree re-weighting (TRW) variational method for upper bounds, the iterated join-graph propagation (IJGP), and Gibbs sampling; see Section 2 for a very brief discussion of these approaches. For weight learning, we compare against the Alchemy system. Unless otherwise specified, the energy function used is always the number of violated constraints, and we use a $50\%$ ratio of random moves ($p = 0.5$). The algorithm is run for 20 iterations, with an initial modification factor $F_0 = 1.5$. The experiments were conducted on a 16-core 2.4 GHz Intel Xeon machine with 32 GB memory, running RedHat Linux.

**Hard constraints.**   First, consider models with only hard constraints, which define a uniform measure on the set of satisfying assignments. In this case, the problem of computing the partition function is equivalent to standard model counting. We compare the performance of `FocusedFlatSAT` with `MCMC-FlatSat` and with two state-of-the-art approximate model counters: SampleCount [13] and SampleMiniSATExact [14]. The instances used are taken from earlier work [11]. The results in Table 1 show that `FocusedFlatSAT` almost always obtains much more accurate solution counts, and is often significantly faster (about an order of magnitude faster than `MCMC-FlatSat`).

**Soft Constraints.**   We consider Ising Models defined on an $n \times n$ square lattice where $P(\sigma) = \sum_\sigma \exp(-E(\sigma))$ with $E(\sigma) = \sum_{(i,j)} w_{ij} \mathcal{I}[\sigma_i \neq \sigma_j]$. Here $\mathcal{I}$ is the indicator function. This imposes a penalty $w_{ij}$ if spins $\sigma_i$ and $\sigma_j$ are not aligned. We consider a ferromagnetic case where $w_{ij} = w > 0$ for all edges, and a frustrated case with a mixture of positive and negative interactions.

The partition function for these planar models is computable with a specialized polynomial time algorithm, as long as there is no external magnetic field [2]. In Figure 3, we compare the true value of the partition function $Z^*$ with the estimate obtained using `FocusedFlatSAT` and with the upper

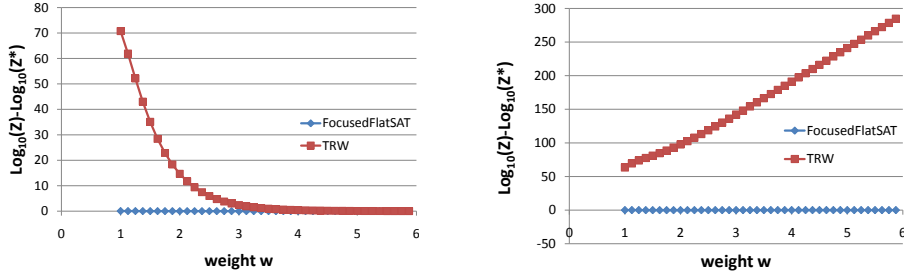

Figure 3: Error in $\log_{10}(Z)$. Left: $40 \times 40$ ferromagnetic grid. Right: $32 \times 32$ spin glass grid.

Table 2: Log partition function for weighted formulas.

| Instance | n | m | Weight | $\log_{10} Z(w)$ | FocusedFlatSat $\log_{10} Z(w)$ | Time | IJGP-SampleSearch $\log_{10} Z(w)$ | Time | Gibbs $\log_{10} Z(w)$ | Time |
|---|---|---|---|---|---|---|---|---|---|---|
| grid32x32 | 1024 | 3968 | 1 | 16.0920 | **16.0964** | **628** | 14.4330 | 600 | 15.4856 | 651 |
| grid32x32 | 1024 | 3968 | 1 | 16.0920 | **16.0964** | **628** | 13.8980 | 2000 | | |
| grid40x40 | 1600 | 6240 | 1 | 23.5434 | **23.4844** | **1522** | 15.9386 | 2000 | 22.3125 | 1650 |
| 2bitmax6 | 252 | 766 | 5 | $> 29.3222$ | **30.4373** | **360** | 12.0526 | 600 | 25.1274 | 732 |
| 2bitmax6 | 252 | 766 | 5 | $> 29.3222$ | **30.4373** | **360** | 12.3802 | 2000 | | |
| wff.100.150 | 100 | 150 | 5 | $> 21.2553$ | 21.3187 | 5 | 21.3373 | 200 | 21.3992 | 40 |
| wff.100.150 | 100 | 150 | 8 | $> 21.2553$ | 21.2551 | 5 | 21.2694 | 200 | 21.3107 | 40 |
| ls8-normalized | 301 | 1603 | 3 | $> 11.7324$ | 17.6655 | 589 | 16.5458 | 600 | 8.6825 | 708 |
| ls8-normalized | 301 | 1603 | 6 | $> 11.7324$ | **11.7974** | **589** | -2.3987 | 600 | -17.356 | 770 |
| ls8-normalized | 301 | 1603 | 6 | $> 11.7324$ | **11.7974** | **589** | -1.7459 | 1200 | | |
| ls8-normalized | 301 | 1603 | 6 | $> 11.7324$ | **11.7974** | **589** | -1.8578 | 2000 | | |
| ls8-simplified-2 | 172 | 673 | 6 | $> 4.3083$ | **4.3379** | **100** | -1.8305 | 1200 | 2.8516 | 300 |
| ls8-simplified-4 | 119 | 410 | 6 | $> 2.2479$ | 2.3399 | 63 | 2.7037 | 1200 | -6.7132 | 174 |
| ls8-simplified-5 | 83 | 231 | 6 | $> 1.3424$ | 1.3880 | 40 | 1.3688 | 600 | 1.3420 | 51 |

bound given by TRW (which is generally much faster but inaccurate), for a range of $w$ values. What is plotted is the accuracy, $\log Z - \log Z^*$. We see that the estimate provided by FocusedFlatSAT is very accurate throughout the range of $w$ values. For the ferromagnetic model, the bounds obtained by TRW, on the other hand, are tight only when the weights are sufficiently high, when essentially only the two ground states of energy zero matter. On spin glasses, where computing ground states is itself an intractable problem, TRW is unsurprisingly inaccurate even in the high weights regime. The consistent accuracy of FocusedFlatSAT here is a strong indication that the method is accurately computing the density of most of the underlying states. This is because, as the weight $w$ changes, the value of the partition function is dominated by the contributions of a different set of states.

Table 2 (top) shows a comparison with IJGP-SampleSearch and Gibbs Sampling for the ferromagnetic case with $w = 1$. Here FocusedFlatSAT provides the most accurate estimates, even when other methods are given a longer running time. E.g., IJGP is two orders of magnitude off for the $32 \times 32$ grid.[3] Results with other weights are similar but omitted due to limited space. FocusedFlatSAT also significantly outperforms IJGP and Gibbs sampling in accuracy on the circuit synthesis instance *2bitmax6*. All methods perform well on randomly generated 3-SAT instances, but FocusedFlatSAT is much faster.

As another test case, we use formulas from a previously used model counting benchmark involving $n \times n$ Latin Square completion [11], and add a weight $w$ to each constraint. Since these instances have high treewidth, are non-planar, and beyond the reach of direct enumeration, we don't have ground truth for this benchmark. However, we are able to provide a lower bound,[4] which is given by the number of models of the original formula. Our results are reported in Table 2. Our lower bound indicates that the estimate given by FocusedFlatSAT is more accurate, even when other methods are given a longer running time. As the last 3 lines of the table show, IJGP and Gibbs sampling improve in performance as the problem is simplified more and more, by fixing the values of 2, 4, or 5 "cells" and simplifying the instance. Nonetheless, on the un-simplified *ls8-normalized* with weight 6, both IJGP and Gibbs sampling underestimate by over 12 orders of magnitude.

Table 3: Weight learning: likelihood of the training data $\overline{x}$ computed using learned weights.

| Type | Training Data | Optimal Likelihood (**O**) | FocusedFlatSAT Accuracy (**F/O**) | Alchemy Accuracy (**A/O**) |
|---|---|---|---|---|
| ThreeChain(30) | $\overline{x} =$ data-30-1 | $4.09 \times 10^{-27}$ | **1.0** | 0.08 |
| | $\overline{x} =$ data-30-2 | $9.31 \times 10^{-10}$ | **1.0** | 0.93 |
| FourChain(5) | $\overline{x} =$ dataFC-5-1 | $5.77 \times 10^{-6}$ | **1.0** | 0.61 |
| | $\overline{x} =$ dataFC-5-2 | $3.84 \times 10^{-3}$ | **1.0** | 0.000097 |
| HChain(10) | $\overline{x} =$ dataH-10-1 | $1.19 \times 10^{-9}$ | **1.0** | 0.87 |
| | $\overline{x} =$ dataH-10-2 | $2.62 \times 10^{-9}$ | **1.0** | 0.53 |
| SocialNetwork(5) | $\overline{x} =$ data-SN-1 | $2.98 \times 10^{-8}$ | **1.0** | 0.69 |
| | $\overline{x} =$ data-SN-2 | $2.44 \times 10^{-9}$ | **1.0** | 0.2 |

**Weight learning.** Suppose the set of soft constraints $\mathcal{C}_{\text{soft}}$ is composed of $M$ disjoint sets of constraints $\{\mathcal{S}_i\}_{i=1}^M$, where all the constraints $c \in \mathcal{S}_i$ have the same weight $w_i$ that we wish to learn from data (for instance, these constraints can all be groundings of the same first order formula in Markov Logic [8]). Let us assume for simplicity that there are no hard constraints. The probability $P_{\mathbf{w}}(x)$ can be parameterized by a weight vector $\mathbf{w} = (w_1, \ldots, w_M)$. The key observation is that the partition function can be written as $Z(\mathbf{w}) = \sum_{\ell_1} \sum_{\ell_2} \cdots \sum_{\ell_M} n(\ell_1, \ldots, \ell_M) \exp(-\mathbf{w} \cdot \ell)$, where $n(\ell_1, \ldots, \ell_M)$ gives the number of configurations that violate $\ell_i$ constraints of type $\mathcal{S}_i$ for $i = 1, \ldots, M$. This function $n(\ell_1, \ldots, \ell_M)$ is precisely the density of states required to compute $Z(\mathbf{w})$ for all values of $\mathbf{w}$, without additional inference steps.

Given training data $\overline{x} \in \{0, 1\}^N$, the problem of weight learning is that of finding $\arg\max_{\mathbf{w}} P_{\mathbf{w}}(\overline{x})$ where $P_{\mathbf{w}}(\overline{x})$ is given by Eqn. (1). Once we compute $n(\ell_1, \ldots, \ell_M)$ using FocusedFlatSAT, we can efficiently evaluate $Z(\mathbf{w})$, and therefore $P_{\mathbf{w}}(\overline{x})$, as a function of the parameters $\mathbf{w} = (w_1, \ldots, w_M)$. Using this efficient evaluation as a black-box, we can solve the weight learning problem using a numerical optimization package with no additional inference steps required.[5]

We evaluate this learning method on relatively simple instances on which commonly used software such as Alchemy can be a few orders of magnitude off from the optimal likelihood of the training data. Specifically, Table 3 compares the likelihood of the training data under the weights learned by FocusedFlatSAT and by Generative Weight Learning [7], as implemented in Alchemy, for four types of Markov Logic theories. The Optimal Likelihood value is obtained using a partition function computed either by direct enumeration or using analytic results for the synthetic instances.

The instance *ThreeChain(K)* is a grounding of the following first order formulas $\forall x P(x) \Rightarrow Q(x), \forall x Q(x) \Rightarrow R(x), \forall x R(x) \Rightarrow P(x)$ while *FourChain(K)* is a similar chain of 4 implications. The instance *HChain(K)* is a grounding of $\forall x P(x) \wedge Q(x) \Rightarrow R(x), \forall x R(x) \Rightarrow P(x)$ where $x \in \{a_1, a_2, \ldots, a_K\}$. The instance *SocialNetwork(K)* (from the Alchemy Tutorial) is a grounding of the following first order formulas where $x, y \in \{a_1, a_2, \ldots, a_K\}$: $\forall x \, \forall y \, Friend(x, y) \Rightarrow (Smokes(x) \Leftrightarrow Smokes(y)), \forall x \, Smokes(x) \Rightarrow Cancer(x)$.

Table 3 shows the accuracy of FocusedFlatSAT and Alchemy for the weight learning task, as measured by the resulting likelihood of observing the data in the learned model, which we are trying to maximize. The accuracy is measured as the ratio of the optimal likelihood (O) and the likelihood in the learned model (F and A, resp.). In these instances, FocusedFlatSAT always matches the optimal likelihood up to two digits of precision, while Alchemy can underestimate it by several orders of magnitude, e.g., by over 4 orders in the case of *FourChain(5)*.

## 6 Conclusion

We introduced FocusedFlatSAT, a Markov Chain Monte Carlo technique based on the flat histogram method with a random walk style component to estimate the partition function from the density of states. We demonstrated the effectiveness of our approach on several types of problems. Our method outperforms the current state-of-the-art techniques on a variety of instances, at times by several orders of magnitude. Moreover, from the density of states we can obtain directly the partition function $Z(w)$ as a function of the model parameters $w$. We show an application of this property to weight learning in Markov Logic Networks.

## Footnotes

*Supported by NSF Expeditions in Computing award for Computational Sustainability (grant 0832782).

[1]The chain is finite, irreducible, and aperiodic, therefore ergodic.

[2]Benchmark instances available online at `http://www.cs.cornell.edu/~ermonste`

[3]On smaller instances with limited treewidth, IJGP-SampleSearch quickly provides good estimates.

[4]The upper bound provided by TRW is very loose on this benchmark (possibly because of the conversion to a pairwise field) and not reported.

[5]Storing the full density function $n(\ell_1, \ldots, \ell_M)$ of course requires space (and hence time) that is exponential in $M$. One must use a relatively coarse partitioning of the state space for scalability when $M$ is large.

# References

[1] Martin J Wainwright and Michael I Jordan. *Graphical Models, Exponential Families, and Variational Inference*. Now Publishers Inc., Hanover, MA, USA, 2008.

[2] N.N. Schraudolph and D. Kamenetsky. Efficient exact inference in planar Ising models. In *Proc. of NIPS-08*, 2008.

[3] F. Wang and DP Landau. Efficient, multiple-range random walk algorithm to calculate the density of states. *Physical Review Letters*, 86(10):2050–2053, 2001.

[4] M.J. Wainwright, T.S. Jaakkola, and A.S. Willsky. A new class of upper bounds on the log partition function. *Information Theory, IEEE Transactions on*, 51(7):2313–2335, 2005.

[5] Vibhav Gogate and Rina Dechter. SampleSearch: A Scheme that Searches for Consistent Samples. *Journal of Machine Learning Research*, 2:147–154, 2007.

[6] Mark Jerrum and Alistair Sinclair. *The Markov chain Monte Carlo method: an approach to approximate counting and integration*, pages 482–520. PWS Publishing Co., Boston, MA, USA, 1997.

[7] P. Domingos, S. Kok, H. Poon, M. Richardson, and P. Singla. Unifying logical and statistical ai. In *Proc. of AAAI-06*, pages 2–7, Boston, Massachusetts, 2006. AAAI Press.

[8] M. Richardson and P. Domingos. Markov logic networks. *Machine Learning*, 62(1):107–136, 2006.

[9] H. Poon and P. Domingos. Sound and efficient inference with probabilistic and deterministic dependencies. In *Proc. of AAAI-06*, pages 458–463, 2006.

[10] J.S. Yedidia, W.T. Freeman, and Y. Weiss. Constructing free-energy approximations and generalized belief propagation algorithms. *Information Theory, IEEE Transactions on*, 51(7):2282–2312, 2005.

[11] S. Ermon, C. Gomes, and B. Selman. Computing the density of states of Boolean formulas. In *Proc. of CP-2010*, 2010.

[12] B. Selman, H.A. Kautz, and B. Cohen. Local search strategies for satisfiability testing. In *DIMACS Series in Discrete Mathematics and Theoretical Computer Science*, 1996.

[13] C.P. Gomes, J. Hoffmann, A. Sabharwal, and B. Selman. From sampling to model counting. In *Proc. of IJCAI-07*, 2007.

[14] V. Gogate and R. Dechter. Approximate counting by sampling the backtrack-free search space. In *Proc. of AAAI-07*, pages 198–203, 2007.

